# White Functionals for Anomaly Detection in Dynamical Systems

**Marco Cuturi**
ORFE - Princeton University
mcuturi@princeton.edu

**Jean-Philippe Vert**
Mines ParisTech, Institut Curie, INSERM U900
Jean-Philippe.Vert@mines.org

**Alexandre d'Aspremont**
ORFE - Princeton University
aspremon@princeton.edu

## Abstract

We propose new methodologies to detect anomalies in discrete-time processes taking values in a probability space. These methods are based on the inference of functionals whose evaluations on successive states visited by the process are stationary and have low autocorrelations. Deviations from this behavior are used to flag anomalies. The candidate functionals are estimated in a subspace of a reproducing kernel Hilbert space associated with the original probability space considered. We provide experimental results on simulated datasets which show that these techniques compare favorably with other algorithms.

## 1 Introduction

Detecting abnormal points in small and simple datasets can often be performed by visual inspection, using notably dimensionality reduction techniques. However, non-parametric techniques are often the only credible alternative to address these problems on the many high-dimensional, richly structured data sets available today.

When carried out on *independent and identically distributed* (i.i.d) observations, anomaly detection is usually referred to as outlier detection and is in many ways equivalent to density estimation. Several density estimators have been used in this context and we refer the reader to the exhaustive review in [1]. Among such techniques, methods which estimate non-parametric alarm functions in reproducing kernel Hilbert spaces (rkHs) are particularly relevant to our work. They form alarm functions of the type $f(\cdot) = \sum_{i \in I} c_i k(x_i, \cdot)$, where $k$ is a positive definite kernel and $(c_i)_{i \in I}$ is a family of coefficients paired with a family $(x_i)_{i \in I}$ of previously observed data points. A new observation $x$ is flagged as anomalous whenever $f(x)$ goes outside predetermined boundaries which are also provided by the algorithm. Two well known kernel methods have been used so far for this purpose, namely kernel principal component analysis (kPCA) [2] and one-class support vector machines (ocSVM) [3]. The ocSVM is a popular density estimation tool and it is thus not surprising that it has already found successful applications to detect anomalies in i.i.d data [4]. kPCA can also be used to detect outliers as described in [5], where an outlier is defined as any point far enough from the boundaries of an ellipsoid in the rkHs containing most of the observed points.

These outlier detection methods can also be applied to *dynamical systems*. We now monitor discrete time stochastic processes $Z = (Z_t)_{t \in \mathbb{N}}$ taking values in a space $\mathcal{Z}$ and, based on previous observations $z_{t-1}, \cdots, z_0$, we seek to detect whether a new observation $z_t$ abnormally deviates from the usual dynamics of the system. As explained in [1], this problem can be reduced to density estimation when either $Z_t$ or a suitable representation of $Z_t$ that includes a finite number of lags is Markovian, i.e. when the conditional probability of $Z_t$ given its past depends only on the values taken by $Z_{t-1}$.

In practice, anomaly detection then involves a two step procedure. It first produces an estimator $\hat{Z}_t$ of the conditional expectation of $Z_t$ given $Z_{t-1}$ to extract an empirical estimator for the residues $\hat{\varepsilon}_t = Z_t - \hat{Z}_t$. Under an i.i.d assumption, abnormal residues can then be used to flag anomalies. This approach and advanced extensions can be used both for multivariate data [6, 7] and linear processes in functional spaces [8] using spaces of Hölderian functions.

The main contribution of our paper is to propose an estimation approach of alarm functionals that can be used on arbitrary Hilbert spaces and which bypasses the estimation of residues $\hat{\varepsilon}_t \in \mathcal{Z}$ by focusing directly on suitable properties for alarm functionals. Our approach is based on the following intuition. Detecting anomalies in a sequence generated by white noise is a task which is arguably easier than detecting anomalies in arbitrary time-series. In this sense, we look for functionals $\alpha$ such that $\alpha(Z_t)$ exhibits a stationary behavior with low autocorrelations, ideally white noise, which can be used in turn to flag an anomaly whenever $\alpha(Z_t)$ departs from normality. We call functionals $\alpha$ that strike a good balance between exhibiting a *low autocovariance of order 1* and a *high variance* on successive values $Z_t$ a *white functional* of the process $Z$. Our definition can be naturally generalized to higher autocovariance orders as the reader will naturally see in the remaining of the paper.

Our perspective is directly related to the concept of cointegration (see [9] for a comprehensive review) for multivariate time series, extensively used by econometricians to study equilibria between various economic and financial indicators. For a multivariate stochastic process $X = (X_t)_{t \in \mathbb{Z}}$ taking values in $\mathbb{R}^d$, $X$ is said to be cointegrated if there exists a vector $a$ of $\mathbb{R}^d$ such that $(a^T X_t)_{t \in \mathbb{Z}}$ is stationary. Economists typically interpret the weights of $a$ as describing a stable *linear* relationship between various (non-stationary) macroeconomic or financial indicators. In this work we discard the immediate interpretability of the weights associated with *linear* functionals $a^T X_t$ to focus instead on functionals $\alpha$ in a rkHs $\mathcal{H}$ such that $\alpha(Z_t)$ is stationary, and use this property to detect anomalies.

The rest of this paper is organized as follows. In Section 2, we study different criterions to measure the autocorrelation of a process, directly inspired by min/max autocorrelation factors [10] and the seminal work of Box-Tiao [11] on cointegration. We study the asymptotic properties of finite sample estimators of these criterions in Section 3 and discuss the practical estimation of white functionals in Section 4. We discuss relationships with existing methods in Section 5 and provide experimental results to illustrate the effectiveness of these approaches in Section 6.

## 2   Criterions to define white functionals

Consider a process $Z = (Z_t)_{t \in \mathbb{Z}}$ taking values in a probability space $\mathcal{Z}$. $Z$ will be mainly considered in this work under the light of its mapping onto a rkHs $\mathcal{H}$ associated with a bounded and continuous kernel $k$ on $\mathcal{Z} \times \mathcal{Z}$. $Z$ is assumed to be second-order stationary, that is the densities $p(Z_t = z)$ and joint densities $p(Z_t = z, Z_{t+k} = z')$ for $k \in \mathbb{N}$ are independent of $t$. Following [12, 13] we write

$$\phi_t = \varphi(Z_t) - \mathbb{E}_p[\varphi(Z_t)],$$

for the centered projection of $Z$ in $\mathcal{H}$, where $\varphi : z \in \mathcal{Z} \to k(z, \cdot) \in \mathcal{H}$ is the feature map associated with $k$. For two elements $\alpha$ and $\beta$ of $\mathcal{H}$ we write $\alpha \otimes \beta$ for their tensor product, namely the linear map of $\mathcal{H}$ onto itself such that $\alpha \otimes \beta : x \to \langle \alpha, x \rangle_{\mathcal{H}} \ \beta$. Using the notations of [14] we write

$$C = \mathbb{E}_p[\phi_t \otimes \phi_t], \qquad D = \mathbb{E}_p[\phi_t \otimes \phi_{t+1}],$$

respectively for the covariance and autocovariance of order 1 of $\phi_t$. Both $C$ and $D$ are linear operators of $\mathcal{H}$ by weak stationarity [14, Definition 2.4] of $(\phi_t)_{t \in \mathbb{Z}}$, which can be deduced from the second-order stationarity of $Z$. The following definitions introduce two criterions which quantify how related two successive evaluations of $\alpha(Z_t)$ are.

**Definition 1** (**Autocorrelation Factor [10]**). *Given an element $\alpha$ of $\mathcal{H}$ such that $\langle \alpha, C\alpha \rangle_{\mathcal{H}} > 0$, $\gamma(\alpha)$ is the absolute autocorrelation of $\alpha(Z)$ of order 1,*

$$\gamma(\alpha) = |\operatorname{corr}(\alpha(Z_t), \alpha(Z_{t+1})| = \frac{|\langle \alpha, D\alpha \rangle_{\mathcal{H}}|}{\langle \alpha, C\alpha \rangle_{\mathcal{H}}}. \tag{1}$$

The condition $\langle \alpha, C\alpha \rangle_{\mathcal{H}} > 0$ requires that $\operatorname{var} \alpha(\phi_t)$ is not zero, which excludes constant or vanishing functions on the support of the density of $\phi_t$. Note also that defining $\gamma$ requires no other assumption than second-order stationarity of $Z$.

If we assume further that $\phi$ is an autoregressive Hilbertian process of order 1 [14], ARH(1) for short, there exists a compact operator $\rho : \mathcal{H} \rightarrow \mathcal{H}$ and a $\mathcal{H}$ strong white noise[1] $(\varepsilon_t)_{t \in \mathbb{Z}}$ such that

$$\phi_{t+1} = \rho\,\phi_t + \varepsilon_t.$$

In their seminal work, Box and Tiao [11] quantify the predictability of the linear functionals of a vector autoregressive process in terms of variance ratios. The following definition is a direct adaptation of that principle to autoregressive processes in Hilbert spaces. From [14, Theorem 3.2] we have that $C = \rho\,C\rho^* + C_\varepsilon$ where for any linear operator $A$ of $\mathcal{H}$, $A^*$ is its adjoint.

**Definition 2** (**Predictability in the Box-Tiao sense** [11]). *Given an element $\alpha$ of $\mathcal{H}$ such that $\langle \alpha, C\alpha \rangle_{\mathcal{H}} > 0$, the predictability $\lambda(\alpha)$ is the quotient*

$$\lambda(\alpha) = \frac{\mathrm{var}\langle \alpha, \rho\,\phi_t \rangle_{\mathcal{H}}}{\mathrm{var}\langle \alpha, \phi_t \rangle_{\mathcal{H}}} = \frac{\langle \alpha, \rho\,C\,\rho^*\alpha \rangle_{\mathcal{H}}}{\langle \alpha, C\alpha \rangle_{\mathcal{H}}} = \frac{\langle \alpha, DC^{-1}D^*\alpha \rangle_{\mathcal{H}}}{\langle \alpha, C\alpha \rangle_{\mathcal{H}}}. \tag{2}$$

The right hand-side of Equation (2) follows from the fact that $\rho\,C = D$ and $\rho^* = C^{-1}D^*$ [14], the latter equality being always valid irrelevant of the existence of $C^{-1}$ on the whole of $\mathcal{H}$ as noted in [15]. Combining these two equalities gives $\rho\,C\rho^* = DC^{-1}D^*$.

Both $\gamma$ and $\lambda$ are convenient ways to quantify for a given function $f$ of $\mathcal{H}$ the independence of $f(Z_t)$ with its immediate past. We provide in this paragraph a common representation for $\lambda$ and $\gamma$. For any linear operator $A$ of $\mathcal{H}$ and any non-zero element $x$ of $\mathcal{H}$ write $\mathcal{R}(A, x)$ for the Rayleigh quotient

$$\mathcal{R}(A, x) = \frac{\langle x, Ax \rangle_{\mathcal{H}}}{\langle x, x \rangle_{\mathcal{H}}}.$$

We use the notations in [12] and introduce the normalized cross-covariance (or rather auto-covariance in the context of this paper) operator $V = C^{-\frac{1}{2}}DC^{-\frac{1}{2}}$. Note that for any skew-symmetric operator $A$, that is $A = -A^*$, we have that $\langle x, Ax \rangle_{\mathcal{H}} = \langle A^*x, x \rangle_{\mathcal{H}} = -\langle Ax, x \rangle_{\mathcal{H}} = 0$ and thus $\mathcal{R}(A, x) = \mathcal{R}(\frac{A+A^*}{2}, x)$. Both $\lambda$ and $\gamma$ applied on a function $\alpha \in \mathcal{H}$ can thus be written as

$$\gamma(\alpha) = \left| \mathcal{R}\left( \frac{V + V^*}{2}, C^{\frac{1}{2}}\alpha \right) \right|, \quad \lambda(\alpha) = \mathcal{R}(VV^*, C^{\frac{1}{2}}\alpha).$$

As detailed in Section 4, our goal is to estimate functions in $\mathcal{H}$ from data such that they have either low $\gamma$ or $\lambda$ values. Minimizing $\lambda$ is equivalent to solving a generalized eigenvalue problem through the Courant-Fisher-Weyl theorem. Minimizing $\gamma$ is a more challenging problem since the operator $V + V^*$ is not necessarily positive definite. The S-lemma from control theory [16, Appendix B.2] can be used to cast the problem of estimating functions with low $\gamma$ as a semi-definite program. In practice the eigen-decomposition of $V + V^*$ provides good approximate answers.

The formulation of $\gamma$ and $\lambda$ as Rayleigh quotients is also useful to obtain the asymptotic convergence of their empirical counterparts (Section 3) and to draw comparisons with kernel-CCA (Section 5).

## 3  Asymptotics and matrix expressions for empirical estimators of $\gamma$ and $\lambda$

### 3.1  Asymptotic convergence of the normalized cross-covariance operator $V$

The covariance operator $C$ and cross-covariance operator $D$ can be estimated through a finite sample of points $z_0, \cdots, z_n$ translated into a sample of centered points $\phi_1, \cdots, \phi_n$ in $\mathcal{H}$, where $\phi_i = \varphi(z_i) - \frac{1}{n+1}\sum_{j=0}^{n}\varphi(z_j)$. We write

$$C_n = \frac{1}{n-1}\sum_{i=1}^{n}\phi_i \otimes \phi_i, \quad D_n = \frac{1}{n-1}\sum_{i=1}^{n-1}\phi_i \otimes \phi_{i+1},$$

for the estimates of $C$ and $D$ respectively which converge in Hilbert-Schmidt norm [14]. Estimators for $\gamma$ or $\lambda$ require approximating $C^{-\frac{1}{2}}$, which is a typical challenge encountered when studying

ARH(1) processes and more generally stationary linear processes in Hilbert spaces [14, Section 8]. This issue is addressed in this section through a Tikhonov-regularization, that is considering a sequence of positive numbers $\epsilon_n$ we write

$$V_n = (C_n + \epsilon_n I)^{-\frac{1}{2}} D_n (C_n + \epsilon_n I)^{-\frac{1}{2}},$$

for the empirical estimate of $V$ regularized by $\epsilon_n$. We have already assumed that $k$ is bounded and continuous. The convergence of $V_n$ to $V$ in norm is ensured under the additional conditions below

**Theorem 3.** *Assume that $V$ is a compact operator,* $\lim_{n\to\infty} \epsilon_n = 0$ *and* $\lim_{n\to\infty} \frac{(\log n/n)^{\frac{1}{3}}}{\epsilon_n} = 0$. *Then writing $\|\cdot\|_{\mathcal{S}}$ for the Hilbert-Schmidt operator norm,* $\lim_{n\to\infty} \|V_n - V\|_{\mathcal{S}} = 0$.

*Proof.* The structure of the proof is identical to that of of [12, Theorem 1] except that the i.i.d assumption does not hold here. In [12], the norm $\|V_n - V\|_{\mathcal{S}}$ is upper-bounded by the two terms $\|V_n - (C + \epsilon_n I)^{-\frac{1}{2}} D (C + \epsilon_n I)^{-\frac{1}{2}}\|_{\mathcal{S}} + \|(C + \epsilon_n I)^{-\frac{1}{2}} D (C + \epsilon_n I)^{-\frac{1}{2}} - V\|_{\mathcal{S}}$. The second term converges under the assumption that $\epsilon_n \to 0$ [12, Lemma 7] while the first term decreases at a rate that is proportional to the rates of $\|C_n - C\|_{\mathcal{S}}$ and $\|D_n - D\|_{\mathcal{S}}$. With the assumptions above [14, Corollary4.1,Theorem 4.8] gives us that $\|C_n - C\|_{\mathcal{S}} = O((\frac{\log n}{n})^{\frac{1}{2}})$ and $\|D_n - D\|_{\mathcal{S}} = O((\frac{\log n}{n})^{\frac{1}{2}})$. We use this result to substitute the latter rate to the faster rate obtained for i.i.d observations in [12, Lemma 5] and conclude the proof. □

## 3.2 Empirical estimators and matrix expressions

Given $\alpha \in \mathcal{H}$, consider the following estimators of $\gamma(\alpha)$ and $\lambda(\alpha)$ defined in Equations (1) and (2),

$$\gamma_n(\alpha) = \left| \mathcal{R}\left( \frac{V_n + V_n^*}{2}, (C_n + \epsilon_n I)^{\frac{1}{2}} \alpha \right) \right| = \frac{|\langle \alpha, \frac{1}{2}(D_n + D_n^*)\alpha \rangle_{\mathcal{H}}|}{\langle \alpha, (C_n + \epsilon_n I)\alpha \rangle_{\mathcal{H}}},$$

$$\lambda_n(\alpha) = \mathcal{R}(V_n V_n^*, (C_n + \epsilon_n I)^{\frac{1}{2}} \alpha) = \frac{\langle \alpha, D_n(C_n + \epsilon_n I)^{-1} D_n^* \alpha \rangle_{\mathcal{H}}}{\langle \alpha, (C_n + \epsilon_n I)\alpha \rangle_{\mathcal{H}}},$$

which converge to the adequate values through the convergence of $(C_n + \epsilon_n I)^{\frac{1}{2}}$, $V_n + V_n^*$ and $V_n V_n^*$. The $n$ observations $\phi_1, \ldots, \phi_n$ which define the empirical estimators above also span a subspace $\mathcal{H}_n$ in $\mathcal{H}$ which can be used to estimate white functionals. Given $\alpha \in \mathcal{H}_n$ we use any arbitrary decomposition $\alpha = \sum_{i=1}^n \mathbf{a}_i \phi_i$. We write $K$ for the original $n+1 \times n+1$ Gram matrix $[k(z_i, z_j)]_{i,j}$ and $\bar{K}$ for its centered counterpart $\bar{K} = (\mathbf{I}_n - \frac{1}{n}\mathbb{1}_{n,n})K(\mathbf{I}_n - \frac{1}{n}\mathbb{1}_{n,n}) = [\langle \phi_i, \phi_j \rangle_{\mathcal{H}}]_{i,j}$. Because of the centering $\mathrm{span}\{\phi_0, \ldots, \phi_n\}$ is actually equal to $\mathrm{span}\{\phi_1, \ldots, \phi_n\}$ and we will only use the $n \times n$ matrix $\mathbf{K}$ obtained by removing the first row and column of $\bar{K}$.

For a $n \times n$ matrix $M$, we write $M_{-i}$ for the $n \times n-1$ matrix obtained by removing the $i^{\mathrm{th}}$ column of $M$. With these notations, $\lambda_n$ and $\gamma_n$ take the following form when evaluated on $\alpha \in \mathcal{H}_n$,

$$\gamma_n(\alpha) = \gamma_n\left( \sum_{i=1}^n \mathbf{a}_i \phi_i \right) = \frac{1}{2} \frac{|\mathbf{a}^T(\mathbf{K}_{-1}\mathbf{K}_{-n}^T + \mathbf{K}_{-1}\mathbf{K}_{-n}^T)\mathbf{a}|}{\mathbf{a}^T(\mathbf{K}^2 + n\epsilon_n \mathbf{K})\mathbf{a}},$$

$$\lambda_n(\alpha) = \lambda_n\left( \sum_{i=1}^n \mathbf{a}_i \phi_i \right) = \frac{\mathbf{a}^T \mathbf{K}_{-1}\mathbf{K}_{-n}^T(\mathbf{K}^2 + n\epsilon_n \mathbf{K})^{-1}\mathbf{K}_{-n}\mathbf{K}_{-1}^T \mathbf{a}}{\mathbf{a}^T(\mathbf{K}^2 + n\epsilon_n \mathbf{K})\mathbf{a}}.$$

If $\epsilon_n$ follows the assumptions of Theorem 3, both $\gamma_n$ and $\lambda_n$ converge to $\gamma$ and $\lambda$ pointwise in $\mathcal{H}_n$.

## 4 Selecting white functionals in practice

Both $\gamma(\alpha)$ and $\lambda(\alpha)$ are proxies to quantify the independence of successive observations $\alpha(Z_t)$. Namely, functions with low $\gamma$ and $\lambda$ are likely to have low autocorrelations and be stationary when evaluated on the process $Z$, and the same can be said of functions with low $\gamma_n$ and $\lambda_n$ asymptotically. However, when $\mathcal{H}$ is of high or infinite dimension, the direct minimization of $\gamma_n$ and $\lambda_n$ is likely to result in degenerate functions[2] which may have extremely low autocovariance on $Z$ but very low variance as well. We select white functionals having this trade off in mind, such that both $\langle \alpha, C, \alpha \rangle_{\mathcal{H}}$ is not negligible and $\gamma$ or $\lambda$ are low at the same time.

### 4.1 Enforcing a lower bound on $\langle \alpha, C\alpha \rangle_{\mathcal{H}}$

We consider the following strategy: following the approach outlined in [14, Section 8] to estimate autocorrelation operators, and more generally in [17] in the context of kernel methods, we restrict $\mathcal{H}_n$ to the directions spanned by the $p$ first eigenfunctions of the operator $C_n$. Namely, suppose $C_n$ can be decomposed as $C_n = \sum_{i=1}^n g_i e_i \otimes e_i$ where $e_i$ is an orthonormal basis of eigenvectors with eigenvalues in decreasing order $g_1 \geq g_2 \geq \cdots \geq g_n \geq 0$. For $1 \leq p \leq n$ We write $\mathcal{H}_p$ for the span$\{e_1, \ldots, e_p\}$ of the $p$ first eigenfunctions. Any function $\alpha$ in $\mathcal{H}_p$ is such that $\langle \alpha, C_n \alpha \rangle_{\mathcal{H}} \geq g_p$ and thus allows us to keep the empirical variance of $\alpha(Z_t)$ above a certain threshold. Let $\mathbf{E}_p$ be the $n \times p$ coordinate matrix of eigenvectors[3] $e_1, \ldots, e_p$ expressed in the family of $n$ vectors $\phi_1, \ldots, \phi_n$ and $\mathbf{G}$ the $p \times p$ diagonal matrix of terms $(g_1, \ldots, g_p)$. We consider now a function $\beta = \sum_i^p \mathbf{b}_i e_i$ in $\mathcal{H}_p$, and note that

$$\gamma_n(\beta) = \frac{1}{2} \frac{|\mathbf{b}^T \mathbf{E}_p^T (\mathbf{K}_{-1} \mathbf{K}_{-n}^T + \mathbf{K}_{-1} \mathbf{K}_{-n}^T) \mathbf{E}_p \mathbf{b}|}{\mathbf{b}^T (\mathbf{G} + n\epsilon_n \mathbf{I}) \mathbf{b}}, \tag{3}$$

$$\lambda_n(\beta) = \frac{\mathbf{b}^T \mathbf{E}_p^T \mathbf{K}_{-1} \mathbf{K}_{-n}^T (\mathbf{K}^2 + n\epsilon_n \mathbf{K})^{-1} \mathbf{K}_{-n} \mathbf{K}_{-1}^T \mathbf{E}_p \mathbf{b}}{\mathbf{b}^T (\mathbf{G} + n\epsilon_n \mathbf{I}) \mathbf{b}}. \tag{4}$$

We define two different functions of $\mathcal{H}_p$, $\beta_{\mathrm{mac}}$ and $\beta_{\mathrm{BT}}$, as the the functionals in $\mathcal{H}_p$ whose coefficients correspond to the eigenvector with minimal (absolute) eigenvalue of the two Rayleigh quotients of Equations (3) and (4) respectively. We call these functionals the minimum autocorrelation (MAC) and Box-Tiao (BT) functionals of $Z$. Below is a short recapitulation of all the computational steps we have described so far.

- **Input**: $n + 1$ observations $z_0, \cdots, z_n \in \mathcal{Z}$ of a time-series $Z$, a p.d. kernel $k$ on $\mathcal{Z} \times \mathcal{Z}$ and a parameter $p$ (we propose an experimental methodology to set $p$ in Section 6.3)
- **Output**: a real-valued function $f(\cdot) = \sum_{i=0}^n c_i k(z_i, \cdot)$ that is a white functional of $Z$.
- **Algorithm**:
  - Compute the $(n+1) \times (n+1)$ kernel matrix $K$, center it and drop the first row and column to obtain $\mathbf{K}$.
  - Store $\mathbf{K}$'s $p$ first eigenvectors and eigenvalues in matrices $\mathbf{U}$ and $\mathrm{diag}(v_1, \cdots, v_p)$.
  - Compute $\mathbf{E}_p = \mathbf{U} \, \mathrm{diag}(v_1, \cdots, v_p)^{-1/2}$ and $\mathbf{G} = \frac{1}{n} \mathrm{diag}(v_1, \cdots, v_p)$.
  - Compute the matrix numerator $\mathbf{N}$ and denominator $\mathbf{D}$ of either Equation (3) or Equation (4) and recover the eigenvector $\mathbf{b}$ with minimal absolute eigenvalue of the generalized eigenvalue problem $(\mathbf{N}, \mathbf{D})$
  - Set $\mathbf{a} = \mathbf{E}_p \mathbf{b} \in \mathbb{R}^n$. Set $c_0 = -\frac{1}{n} \sum_1^n \mathbf{a}_j$ and $c_i = \mathbf{a}_i - \frac{1}{n} \sum_1^n \mathbf{a}_j$

## 5 Relation to other methods and discussion

The methods presented in this work offer numerous parallels with other kernel methods such as kernel-PCA or kernel-CCA which, similarly to the BT and MAC functionals, provide a canonical decomposition of $\mathcal{H}_n$ into $n$ ranked eigenfunctions.

When $\mathcal{Z}$ is finite dimensional, the authors of [18] perform PCA on a time-series sample $z_0, \ldots, z_n$ and consider its eigenvector with smallest eigenvalue to detect cointegrated relationships in the process $Z_t$. Their assumption is that a linear mapping $\alpha^T Z_t$ that has small variance on the whole sample can be interpreted as an integrated relationship. Although the criterion considered by PCA, namely variance, disregards the temporal structure of the observations and only focuses on the values spanned by the process, this technique is useful to get rid of all non-stationary components of $Z_t$. On the other hand, kernel-PCA [2], a non-parametric extension of PCA, can be naturally applied for anomaly detection in an i.i.d. setting [5]. It is thus natural to use kernel-PCA, namely an eigenfunction with low variance, and hope that it will have low autocorrelation to define white functionals of a process. Our experiments show that this is indeed the case and in agreement with [5] seem to

indicate that the eigenfunctions which lie at the very low end of the spectrum, usually discarded as noise and less studied in the literature, can prove useful for anomaly detection tasks.

kernel-CCA and variations such as NOCCO [12] are also directly related to the BT functional. Indeed, the operator $VV^*$ used in this work to define $\lambda$ is used in the context of kernel-CCA to extract one of the two functions which maximally correlate two samples, the other function being obtained from $V^*V$. Notable differences between our approach and kernel-CCA are: 1. in the context of this paper, $V$ is an *autocorrelation* operator while the authors of [12] consider *normalized covariances* between two different samples; 2. kernel-CCA assumes that samples are independently and identically drawn, which is definitely not the case for the BT functional; 3. while kernel-CCA maximizes the Rayleigh quotient of $VV^*$, we look for eigenfunctions which lie at the lower end of the spectrum of the same operator. A possible extension of our work is to look for two functionals $f$ and $g$ which, rather than maximize the correlation of two distinct samples as is the case in CCA, are estimated to *minimize* the correlation between $g(z_t)$ and $f(z_{t+1})$. This direction has been explored in [19] to shed a new light on the Box-Tiao approach in the finite dimensional case.

# 6 Experimental results using a population dynamics model

## 6.1 Generating sample paths polluted by anomalies

We consider in this experimental section a simulated dynamical system perturbed by arbitrary anomalies. To this effect, we use the Lotka-Volterra equations to generate time-series quantifying the populations of different species competing for common resources. For $S$ species, the model tracks the population level $X_{t,i}$ at time $t$ of each species $i$, which is a number bounded between 0 and 1. Values of 0 and 1 account respectively for the extinction and the saturation levels of each species. Writing $\circ$ for the coordinate-wise kronecker product of vectors and matrices and $h > 0$ for a discretization step, the population vector $X_t \in [0,1]^S$ follows the discrete-time dynamic equation

$$X_{t+1} = X_t + \frac{1}{h} r \circ X_t \circ (\mathbb{1}_S - AX_t).$$

We consider the following coefficients introduced in [20] which are known to yield chaotic behavior,

$$S = 4, \quad r = \begin{pmatrix} 1 \\ 0.72 \\ 1.53 \\ 1.27 \end{pmatrix}, \quad A = \begin{pmatrix} 1 & 1.09 & 1.52 & 0 \\ 0 & 1 & .44 & 1.36 \\ 2.33 & 0 & 1 & .47 \\ 1.21 & .51 & .35 & 1 \end{pmatrix},$$

which can be turned into a stochastic system by adding an i.i.d. standard Gaussian noise $\varepsilon_t$,

$$Z_{t+1} = Z_t + \frac{1}{h} r \circ Z_t \circ (\mathbb{1}_4 - AZ_t) + \sigma_\varepsilon \varepsilon_t. \tag{5}$$

Whenever the equations generate coordinates below 0 or above 1, the violating coordinates are set to $0 + u$ or $1 - u$ respectively, where $u$ is uniform over $[0, 0.01]$.

We consider trajectories of length 800 of the Lotka-Volterra system described in Equation (5). For each experiment we draw a starting point $Z_0$ randomly with uniform distribution on $[0,1]^4$, discard the 10 first iterations and generate 400 iterations following Equation (5). Following this we select randomly (uniformly over the remaining 400 steps) 40 time stamps $t_1, \cdots, t_{40}$ where we introduce a random perturbation at $t_k$ such that $Z_{t_k}$, rather than following the dynamic of Equation (5) is randomly perturbed by a noise $\delta_t$ chosen uniformly over $\{-1, 1\}^4$ with a magnitude $\sigma_\delta$, that is

$$Z_{t_k} = Z_{t_k - 1} + \sigma_\delta \delta_{t_k - 1}.$$

For all other timestamps $t_k < t < t_{k+1}$, the system follows the usual dynamic of Equation (5). Anomalies violate the usual dynamics in two different ways: first, they ignore the usual dynamical equations and the current location of the process to create instead purely random increments; second, depending on the magnitude of $\sigma_\delta$ relative to $\sigma_\epsilon$, such anomalies may induce unusual jumps.

## 6.2 Estimation of white functionals and other alarm functions

We compare in this experiment five techniques to detect the anomalies described above: the Box-Tiao functional and a variant described in the paragraph below, the minimal autocorrelation functional, a one-class SVM and the low-variance functional defined by the $(p + 1)^{\text{th}}$ eigenfunction of

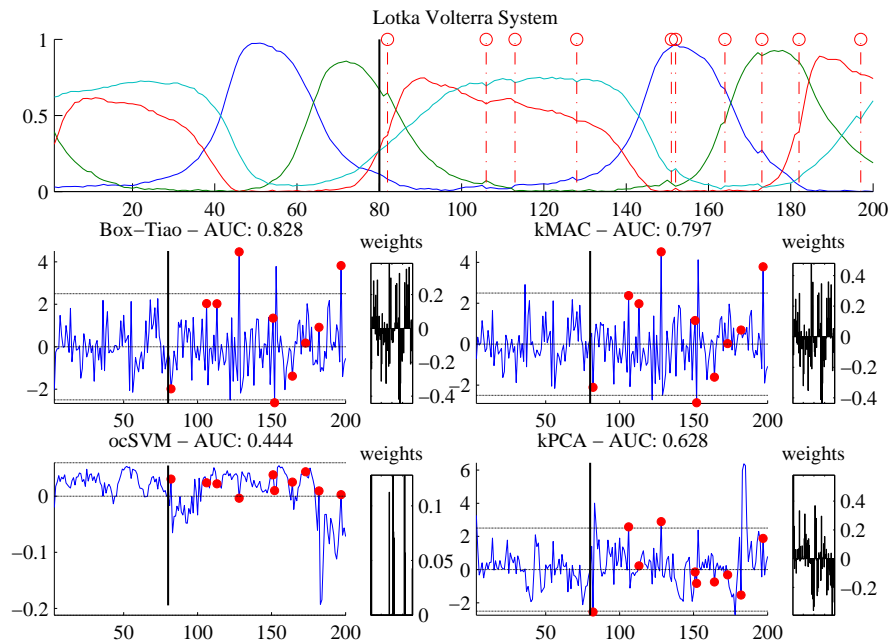

Figure 1: The figure on the top plots a sample path of length 200 of a 4-dimensional Lotka-Volterra dynamic system with perturbations drawn with $\sigma_\varepsilon = .01$ and $\sigma_\delta = 0.02$. The data is split between 80 regular observations and 120 observations polluted by 10 anomalies. All four functionals have been estimated using $\rho = 1$, and we highlight by a red dot the values they take when an anomaly is actually observed. The respective weights associated to each of the 80 training observations are displayed on the right of each methodology.

the empirical covariance $C_n$, given by kernel-PCA. All techniques are parameterized by a kernel $k$. Writing $\Delta z_i = z_i - z_{i-1}$, we use the following mixture of kernels $k$ :

$$k(z_i, z_j) = \rho\, e^{-100\|\Delta z_i - \Delta z_j\|^2} + (1-\rho)e^{-10\|z_i - z_j\|^2}, \tag{6}$$

with $\rho \in [0, 1]$. The first term in $k$ discriminates observations according to their location in $[0, 1]^4$. When $\rho = 0.5$, $k$ accounts for both the state of the system and its most recent increments, while only increments are considered for $\rho = 1$. Anomalies can be detected with both criterions, since they can be tracked down when the process visits unusual regions or undergoes brusque and atypical changes. The kernel widths have been set arbitrarily.

We discuss in this paragraph a variant of the BT functional. While the MAC functional is defined and estimated in order to behave as closely as possible to random i.i.d noise, the BT functional $\beta_{\mathrm{BT}}$ is tuned to be stationary as discussed in [11]. In order to obtain a white functional from $\beta_{\mathrm{BT}}$ it is possible to model the time series $\beta_{\mathrm{BT}}(z_t)$ as an unidimensional autoregressive model, that is estimate (on the training sample again) coefficients $r_1, r_2, \ldots, r_q$ such that

$$\beta_{\mathrm{BT}}(z_t) = \sum_{i=1}^{q} r_i \beta_{\mathrm{BT}}(z_{t-i}) + \hat{\varepsilon}_t^{\mathrm{BT}}.$$

Both the order $q$ and the autoregressive coefficients can be estimated on the training sample with standard AR packages, using for instance Schwartz's criterion to select $q$. Note that although $\phi(Z_t)$ is assumed to be ARH(1), this does not necessarily translate into the fact that the real-valued process $\beta_{\mathrm{BT}}(Z_t) = \langle \beta_{\mathrm{BT}}, \phi_t \rangle_{\mathcal{H}}$ is AR(1) as pointed out in [14, Theorem 3.4]. In practice however we use the residuals $\hat{\varepsilon}_t^{\mathrm{BT}} = \beta_{\mathrm{BT}}(z_t) - \sum_{i=1}^{p} r_i \beta_{\mathrm{BT}}(z_{t-i})$ to define the *Box-Tiao residuals* functional which we write $\tilde{\beta}_{\mathrm{BT}}$.

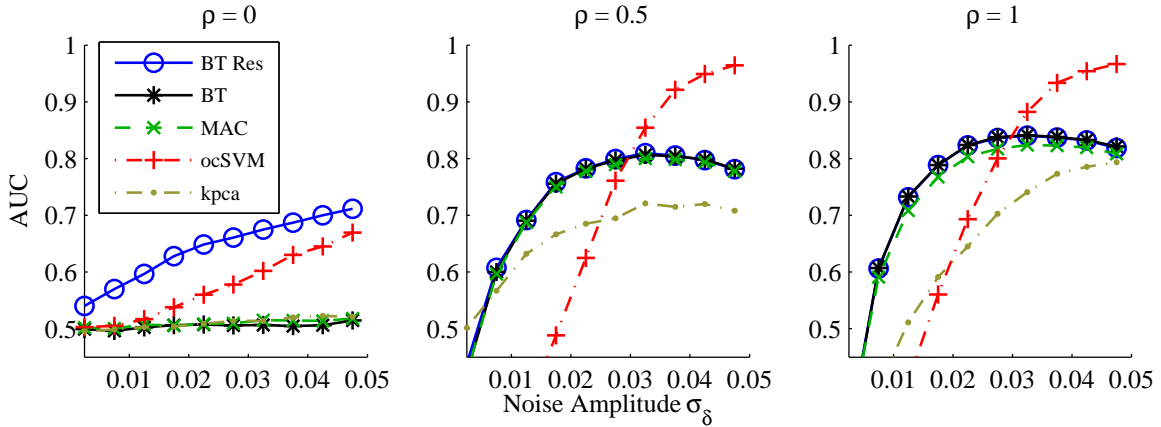

Figure 2: The three successive plot stand for three different values of $\rho = 0, 0.5, 1$. The detection rate naturally increases with the size of the anomaly, to the extent that the task becomes only a gap detection problem when $\sigma_\delta$ becomes closer to $0.05$. Functionals $\beta_{\mathrm{BT}}, \tilde{\beta}_{\mathrm{BT}}$ and $\beta_{\mathrm{mac}}$ have a similar performance and outperform other techniques when the task is most difficult and $\sigma_\delta$ is small.

### 6.3 Parameter selection methodology and numerical results

The BT functional $\beta_{\mathrm{BT}}$ and its residuals $\tilde{\beta}_{\mathrm{BT}}$, the MAC function $\beta_{\mathrm{mac}}$, the one-class SVM $\hat{f}_{\mathrm{ocSVM}}$ and the $p + 1^{\mathrm{th}}$ eigenfunction $e_{p+1}$ are estimated on a set of $400$ observations. We set $p$ through the rule that the $p$ first directions must carry at least 98% of the total variance of $C_n$, that is $p$ is the first integer such that $\sum_{i=1}^{p} g_i > 0.98 \cdot \sum_{i=1}^{n} g_i$. We fix the $\nu$ paramater of the ocSVM to $0.1$. The BT and MAC functionals additionally require the use of a regularization term $\epsilon_n$ which we select by finding the best ridge regressor of $\phi_{t+1}$ given $\phi_t$ through a 4-fold cross validation procedure on the training set. For $\beta_{\mathrm{BT}}, \tilde{\beta}_{\mathrm{BT}}, \beta_{\mathrm{mac}}$ and the kPCA functional $e_{p+1}$ we use their respective empirical mean $\mu$ and variance $\sigma$ on the training set to rescale and whiten their output on the test set, namely consider values $(f(z) - \mu)/\sigma$. Although more elaborate anomaly detection schemes on such unidimensional time-series might be considered, for the sake of simplicity we treat directly these raw outputs as alarm scores.

Having on the one hand the correct labels for anomalies and the scores for all detectors, we vary the threshold at which an alarm is raised to produce ROC curves. We use the area under the curve of each method on each sample path as a performance measure for that path. Figure 1 provides a summary of the performance of each method on a unique sample path of 200 observations and 10 anomalies. Perturbation parameters are set such that $\sigma_\varepsilon = 0.01$ and $\sigma_\delta$ varies between $0.005$ and $0.055$. For each couple $(\sigma_\varepsilon, \sigma_\delta)$ we generate 500 draws and compute the mean AUC of each technique on such draws. We report in Figure 2 these averaged performances for three different choices of the kernel, namely three different values for $\rho$ as defined in Equation (6).

### 6.4 Discussion

In the experimental setting, anomalies can be characterized as unusual increments between two successive states of an otherwise smooth dynamical system. Anomalies are unusual due to their size, controlled by $\sigma_\delta$, and their directions, sampled in $\{-1, 1\}^4$. When the step $\sigma_\delta$ is relatively small, it is difficult to flag correctly an anomaly without taking into account the system's dynamic as illustrated by the relatively poor performance of the ocSVM and the kPCA compared to the BT, BTres and MAC functions. On the contrary, when $\sigma_\delta$ is big, anomalies can be more simply discriminated as big gaps. The methods we propose do not perform as well as the ocSVM in such a setting. We can hypothesize two reasons for this: first, white functionals may be less useful in such a regime that puts little emphasis on dynamics than a simple ocSVM with adequate kernel. Second, in this study the BT and MAC functions flag anomalies whenever an evaluation goes outside of a certain bounding tube. More advanced detectors of a deviation or change from normality, such as CUSUM [21], might be studied in future work.

## Footnotes

[1] namely a sequence $(\varepsilon_t)_{t \in \mathbb{Z}}$ of $\mathcal{H}$ random variables such that (i) $0 < \mathbb{E}\|\varepsilon_t\|^2 = \sigma^2$, $\mathbb{E}\,\varepsilon_t = 0$ and the covariance $C_{\varepsilon_t}$ is constant, equal to $C_\varepsilon$; (ii) $(\varepsilon_t)$ is a sequence of i.i.d $\mathcal{H}$-random variables

[2]Since the rank of operator $V_n$ is actually $n - 1$, we are even guaranteed to find in $\mathcal{H}_n$ a minimizer for $\gamma_n$ and another for $\lambda_n$ with respectively zero predictability and zero absolute autocorrelation.

[3]Recall that if $(\mathbf{u}_i, v_i)$ are eigenvalue and eigenvector pairs of $\mathbf{K}$, the matrix $\mathbf{E}$ of coordinates of eigenfunctions $e_i$ expressed in the $n$ points $\phi_1, \ldots, \phi_n$ can be written as $\mathbf{U} \, \mathrm{diag}(v_i^{-1/2})$ and the eigenvalues $g_i$ are equal to $\frac{v_i}{n}$ if taken in the same order[2].

# References

[1] V. Chandola, A. Banerjee, and V. Kumar. Anomaly detection: A survey. *ACM Computing Surveys*, 2009.

[2] B. Schölkopf, A. Smola, and K. Müller. Nonlinear component analysis as a kernel eigenvalue problem. *Neural Comput.*, 10(5):1299–1319, 1998.

[3] B. Schölkopf, J. C. Platt, J. Shawe-Taylor, A. J. Smola, and R. C. Williamson. Estimating the support of a high-dimensional distribution. *Neural Comput.*, 13:2001, 1999.

[4] A.B. Gardner, A.M. Krieger, G. Vachtsevanos, and B. Litt. One-class novelty detection for seizure analysis from intracranial EEG. *J. Mach. Learn. Res.*, 7:1025–1044, 2006.

[5] H. Hoffmann. Kernel PCA for novelty detection. *Pattern Recognit.*, 40(3):863–874, 2007.

[6] A. J. Fox. Outliers in time series. *J. R. Stat. Soc. Ser. B*, 34(3):350–363, 1972.

[7] R.S. Tsay, D. Pena, and A.E. Pankratz. Outliers in multivariate time series. *Biometrika*, 87(4):789–804, 2000.

[8] A. Laukaitis and A. Račkauskas. Testing changes in Hilbert space autoregressive models. *Lithuanian Mathematical Journal*, 42(4):343–354, 2002.

[9] G. S. Maddala and I. M. Kim. *Unit roots, cointegration, and structural change*. Cambridge Univ. Pr., 1998.

[10] P. Switzer and A.A. Green. Min/max autocorrelation factors for multivariate spatial imagery. *Computer science and statistics*, 16:13–16, 1985.

[11] G. Box and G. C. Tiao. A canonical analysis of multiple time series. *Biometrika*, 64(2):355–365, 1977.

[12] K. Fukumizu, F. R. Bach, and A. Gretton. Statistical consistency of kernel canonical correlation analysis. *J. Mach. Learn. Res.*, 8:361–383, 2007.

[13] A. Berlinet and C. Thomas-Agnan. *Reproducing Kernel Hilbert Spaces in Probability and Statistics*. Kluwer Academic Publishers, 2003.

[14] D. Bosq. *Linear Processes in Function Spaces: Theory and Applications*. Springer, 2000.

[15] A. Mas. Asymptotic normality for the empirical estimator of the autocorrelation operator of an ARH (1) process. *Compt. Rendus Acad. Sci. Math.*, 329(10):899–902, 1999.

[16] S. Boyd and L. Vandenberghe. *Convex Optimization*. Cambridge University Press, 2004.

[17] L. Zwald and G. Blanchard. Finite dimensional projection for classification and statistical learning. *IEEE Trans. Inform. Theory*, 54:4169, 2008.

[18] J. H. Stock and M. W. Watson. Testing for common trends. *J. Am. Stat. Assoc.*, pages 1097–1107, 1988.

[19] P. Bossaerts. Common nonstationary components of asset prices. *J. Econ. Dynam. Contr.*, 12(2-3):347–364, 1988.

[20] J. A. Vano, J. C. Wildenberg, M. B. Anderson, J. K. Noel, and J. C. Sprott. Chaos in low-dimensional lotka-volterra models of competition. *Nonlinearity*, 19(10):2391–2404, 2006.

[21] M. Basseville and I.V Nikiforov. *Detection of abrupt changes: theory and applications*. Prentice-Hall, 1993.

